# Boosting Density Estimation

**Saharon Rosset**
Department of Statistics
Stanford University
Stanford, CA, 94305
*saharon@stat.stanford.edu*

**Eran Segal**
Computer Science Department
Stanford University
Stanford, CA, 94305
*eran@cs.stanford.edu*

## Abstract

Several authors have suggested viewing boosting as a gradient descent search for a good fit in function space. We apply gradient-based boosting methodology to the unsupervised learning problem of density estimation. We show convergence properties of the algorithm and prove that a strength of weak learnability property applies to this problem as well. We illustrate the potential of this approach through experiments with boosting Bayesian networks to learn density models.

## 1 Introduction

Boosting is a method for incrementally building linear combinations of "weak" models, to generate a "strong" predictive model. Given data $\{z_i\}_{i=1}^n$, a basis (or dictionary) of weak learners $\mathcal{H}$ and a loss function $L$, a boosting algorithm sequentially finds models $h_1, h_2, \ldots \in \mathcal{H}$ and constants $\alpha_1, \alpha_2, \ldots \in \mathcal{R}$ to minimize $\sum_i L(z_i, \sum_j \alpha_j h_j(z_i))$. AdaBoost [6], the original boosting algorithm, was specifically devised for the task of classification, where $z_i = (x_i, y_i)$ with $y_i \in \{-1, 1\}$ and $L = L(y_i, \sum_j \alpha_j h_j(x_i))$. AdaBoost sequentially fits weak learners on re-weighted versions of the data, where the weights are determined according to the performance of the model so far, emphasizing the more "challenging" examples. Its inventors attribute its success to the "boosting" effect which the linear combination of weak learners achieves, when compared to their individual performance. This effect manifests itself both in training data performance, where the boosted model can be shown to converge, under mild conditions, to ideal training classification, and in generalization error, where the success of boosting has been attributed to its "separating" — or margin maximizing — properties [18].

It has been shown [8, 13] that AdaBoost can be described as a gradient descent algorithm, where the weights in each step of the algorithm correspond to the gradient of an exponential loss function at the "current" fit. In a recent paper, [17] show that the margin maximizing properties of AdaBoost can be derived in this framework as well. This view of boosting as gradient descent has allowed several authors [7, 13, 21] to suggest "gradient boosting machines" which apply to a wider class of supervised learning problems and loss functions than the original AdaBoost. Their results have been very promising.

In this paper we apply gradient boosting methodology to the unsupervised learning problem of density estimation, using the negative log likelihood loss criterion $L(z, \sum_j \alpha_j h_j(z)) = -\log(\sum_j \alpha_j h_j(z))$. The density estimation problem has been studied extensively in many contexts using various parametric and non-parametric approaches [2, 5]. A particular

framework which has recently gained much popularity is that of Bayesian networks [11], whose main strength stems from their graphical representation, allowing for highly interpretable models. More recently, researchers have developed methods for learning Bayesian networks from data including learning in the context of incomplete data. We use Bayesian networks as our choice of weak learners, combining the models using the boosting methodology. We note that several researchers have considered learning weighted mixtures of networks [14], or ensembles of Bayesian networks combined by model averaging [9, 20].

We describe a generic density estimation boosting algorithm, following the approach of [13]. The main idea is to identify, at each boosting iteration, the basis function $h \in \mathcal{H}$ which gives the largest "local" improvement in the loss at the current fit. Intuitively, $h$ assigns higher probability to instances that received low probability by the current model. A line search is then used to find an appropriate coefficient for the newly selected $h$ function, and it is added to the current model.

We provide a theoretical analysis of our density estimation boosting algorithm, showing an explicit condition, which if satisfied, guarantees that adding a weak learner to the model improves the training set loss. We also prove a "strength of weak learnability" theorem which gives lower bounds on overall training loss improvement as a function of the individual weak learners' performance on re-weighted versions of the training data.

We describe the instantiation of our generic boosting algorithm for the case of using Bayesian networks as our basis of weak learners $\mathcal{H}$ and provide experimental results on two distinct data sets, showing that our algorithm achieves higher generalization on unseen data as compared to a single Bayesian network and one particular ensemble of Bayesian networks. We also show that our theoretical criterion for a weak learner to improve the overall model applies well in practice.

## 2   A density estimation boosting algorithm

At each step $t$ in a boosting algorithm, the model built so far is: $F_{t-1}(z) = \sum_{j<t} \alpha_j h_j(z)$. If we now choose a weak learner $h \in \mathcal{H}$ and add it to our model with a small coefficient $\epsilon$, then developing the training loss of the new model $G = F_{t-1} + \epsilon h$ in a Taylor series around the loss at $F_{t-1}$ gives

$$\sum_i L(z_i, G(z_i)) = \sum_i L(z_i, F_{t-1}(z_i)) + \epsilon \sum_i \frac{\partial L(z_i, F_{t-1}(z_i))}{\partial F_{t-1}(z_i)} h(z_i) + O(\epsilon^2),$$

which in the case of negative log-likelihood loss can be written as

$$\sum_i -log(G(z_i)) = \sum_i -log(F_{t-1}(z_i)) - \epsilon \sum_i \frac{1}{F_{t-1}(z_i)} h(z_i) + O(\epsilon^2).$$

Since $\epsilon$ is small, we can ignore the second order term and choose the next boosting step $h_t$ to maximize $\sum_i \frac{1}{F_{t-1}(z_i)} h(z_i)$. We are thus finding the first order optimal weak learner, which gives the "steepest descent" in the loss at the current model predictions. However, we should note that once $\epsilon$ becomes non-infinitesimal, no "optimality" property can be claimed for this selected $h_t$.

The main idea of gradient-based generic boosting algorithms, such as AnyBoost [13] and GradientBoost [7], is to utilize this first order approach to find, at each step, the weak learner which gives good improvement in the loss and then follow the "direction" of this weak learner to augment the current model. The step size $\alpha_t$ is determined in various ways in the different algorithms, the most popular choice being line-search, which we adopt here.

When we consider applying this methodology to density estimation, where the basis $\mathcal{H}$ is comprised of probability distributions and the overall model $F_t$ is a probability distribution

as well, we cannot simply augment the model, since $F_{t-1} + \alpha_t h_t$ will no longer be a probability distribution. Rather, we consider a step of the form $F_t = (1 - \alpha_t)F_{t-1} + \alpha_t h_t$, where $0 \leq \alpha_t \leq 1$. It is easy to see that the first order theory of gradient boosting and the line search solution apply to this formulation as well.

If at some stage $t$, the current $F_{t-1}$ cannot be improved by adding any of the weak learners as above, the algorithm terminates, and we have reached a global minimum. This can only happen if the derivative of the loss at the current model with respect to the coefficient of each weak learner is non-negative:

$$\forall h \in \mathcal{H}, \quad \frac{\partial \sum_i -\log((1-\alpha)F_{t-1}(z_i) + \alpha h(z_i))}{\partial \alpha}\Big|_{\alpha=0} = n - \sum_i \frac{1}{F_{t-1}}h(z_i) \geq 0.$$

Thus, the algorithm terminates if no $h \in \mathcal{H}$ gives $\sum_i \frac{1}{F_{t-1}}h(z_i) > n$ (see section 3 for proof and discussion).

The resulting generic gradient boosting algorithm for density estimation can be seen in Fig. 1. Implementation details for this algorithm include the choice of the family of weak learners $\mathcal{H}$, and the method for searching for $h_t$ at each boosting iteration. We address these details in Section 4.

---

1. Set $F_0(z)$ to uniform on the domain of $z$
2. For t = 1 to T
       (a) Set $w_i = 1/F_{t-1}(z_i)$
       (b) Find $h_t \in \mathcal{H}$ to maximize $\sum_i w_i h_t(z_i)$
       (c) If $\sum_i w_i h_t(z_i) \leq n$ break.
       (d) Find $\alpha_t = \arg\min_\alpha \sum_i -\log((1-\alpha)F_{t-1}(z_i) + \alpha h_t(z_i))$
       (e) Set $F_t = (1 - \alpha_t)F_{t-1} + \alpha_t h_t$
3. Output the final model $F_T$

---

Figure 1: Boosting density estimation algorithm

## 3 Training data performance

The concept of "strength of weak learnability" [6, 18] has been developed in the context of boosting classification models. Conceptually, this property can be described as follows: "if for any weighting of the training data $\{w_i\}_{i=1}^n$, there is a weak learner $h \in \mathcal{H}$ which achieves weighted training error slightly better than random guessing on the *re*-weighted version of the data using these weights, then the combined boosted learner will have vanishing error on the training data".

In classification, this concept is realized elegantly. At each step in the algorithm, the weighted error of the previous model, using the new weights is exactly $0.5$. Thus, the new weak learner doing "better than random" on the re-weighted data means it can improve the previous weak learner's performance at the current fit, by achieving weighted classification error better than $0.5$. In fact it is easy to show that the weak learnability condition of at least one weak learner attaining classification error less than $0.5$ on the re-weighted data does not hold *only* if the current combined model is the optimal solution in the space of linear combinations of weak learners.

We now derive a similar formulation for our density estimation boosting algorithm. We start with a quantitative description of the performance of the previous weak learner $h_{t-1}$ at the combined model $F_{t-1}$, given in the following lemma:

**Lemma 1** *Using the algorithm of section 2 we get:* $\forall t$, $\sum_i \frac{h_t(z_i)}{F_t(z_i)} = n$, *where $n$ is the number of training examples.*

*Proof*: The line search (step 2(c) in the algorithm) implies:

$$0 \quad = \quad \frac{\partial \sum_i -\log((1-\alpha)F_{t-1}(z_i) + \alpha h_t(z_i))}{\partial \alpha}|_{\alpha=\alpha_t} = \frac{1}{1-\alpha_t}(n - \sum_i \frac{h_t(z_i)}{F_t(z_i)}).$$

Lemma 1 allows us to derive the following stopping criterion (or optimality condition) for the boosting algorithm, illustrating that in order to improve training set loss, the new weak learner only has to exceed the previous one's performance at the current fit.

**Theorem 1** *If there does not exist a weak learner $h \in \mathcal{H}$ such that $\sum_i \frac{1}{F_t(z_i)} h(z_i) > n$, then $F_t$ is the global minimum in the domain of normalized linear combinations of $\mathcal{H}$:*
$$F_t = \arg\min_\gamma \sum_i -log(\sum_j \gamma_j h_j(z_i)) \quad s.t. \; \gamma \geq 0 \; and \; \sum_j \gamma_j = 1$$

*Proof:* This is a direct result of the optimality conditions for a convex function (in this case $-\log$) in a compact domain. $\square$

So unless we have reached the global optimum in the simplex within $span\{\mathcal{H}\}$ (which will generally happen quickly only if $\mathcal{H}$ is very small, i.e. the "weak" learners are very weak), we will have some weak learners doing better than "random" and attaining $\sum_i \frac{h(z_i)}{F_t(z_i)} > n$. If this is indeed the case, we can derive an explicit lower bound for training set loss improvement as a function of the new weak learner's performance at the current model:

**Theorem 2** *Assume:*

1. *The sequence of selected weak learners in the algorithm of section 2 has:* $\sum_i \frac{1}{F_{t-1}(z_i)} h_t(z_i) = n + \lambda_t$

2. $\forall t \; min_i(F_{t-1}(z_i), h_t(z_i)) \geq \epsilon_t$

*Then we get:* $-\sum_i \log(F_t(z_i)) \leq -\sum_i \log(F_{t-1}(z_i)) - \frac{\lambda_t^2 \epsilon_t^2}{2n}$

*Proof:*

$$\frac{\partial \sum_i -\log((1-\alpha)F_{t-1}(z_i) + \alpha h_t(z_i))}{\partial \alpha}|_{\alpha=0} = n - \sum_i \frac{h_t(z_i)}{F_{t-1}(z_i)} = -\lambda_t$$

$$\frac{\partial^2 \sum_i -\log((1-\alpha)F_{t-1}(z_i) + \alpha h_t(z_i))}{\partial \alpha^2} = \sum_i \frac{[F_{t-1}(z_i) - h_t(z_i)]^2}{[(1-\alpha)F_{t-1}(z_i) + \alpha h_t(z_i)]^2} \leq \frac{n}{\epsilon_t^2}$$

Combining these two gives: $\frac{\partial \sum_i -\log((1-\alpha)F_{t-1}(z_i) + \alpha h_t(z_i))}{\partial \alpha n} \leq -\lambda_t + \frac{\alpha n}{\epsilon_t^2}$, which implies:

$$\sum_i \log(F_t(z_i)) - \sum_i \log(F_{t-1}(z_i)) \leq -\int_0^{\frac{\lambda_t \epsilon_t^2}{n}} \lambda_t - \frac{xn}{\epsilon_t^2} dx = -\frac{\lambda_t^2 \epsilon_t^2}{n} + \frac{\lambda_t^2 \epsilon_t^2}{2n} = -\frac{\lambda_t^2 \epsilon_t^2}{2n} \quad \square$$

The second assumption of theorem 2 may not seem obvious but it is actually quite mild. With a bit more notation we could get rid of the need to lower bound $h_t$ completely. For $F_t$, we can see intuitively that a boosting algorithm will not let any observation have exceptionally low probability over time since that would cause this observation to have overwhelming weight in the next boosting iteration and hence the next selected $h_t$ is certain to give it high probability. Thus, after some iterations we can assume that we would actually have a threshold $\epsilon$ independent of the iteration number and hence the loss would decrease at least as the sum of squares of the "weak learnability" quantities $\lambda_t$.

## 4  Boosting Bayesian Networks

We now focus our attention on a specific application of the boosting methodology for density estimation, using Bayesian networks as the weak learners. A Bayesian network is a

graphical model for describing a joint distribution over a set of random variables. Recently, there has been much work on developing algorithms for learning Bayesian networks (both network structure and parameters) from data for the task of density estimation and hence they seem appropriate as our choice of weak learners. Another advantage of Bayesian networks in our context, is the ability to tune the strength of the weak learners using parameters such as number of edges and strength of prior.

Assume we have categorical data $\{z_i\}_{i=1}^n$ in a domain $\mathcal{Z}$ where each of the $n$ observations contains assignments to $k$ variables. We rewrite step 2(b) of the boosting algorithm as:
  $2(b)$ Find $h_t \in \mathcal{H}$ to maximize $\sum_{z \in \mathcal{Z}} v_z h(z)$, where $v_z = \sum_{z_i = z} w_i$
In this formulation, all possible values of $z$ have weights, some of which may be 0.

As mentioned above, the two main implementation-specific details in the generic density estimation algorithm are the set $\mathcal{H}$ of weak models and the method for searching for the "optimal" weak model $h_t$ at each boosting iteration. When boosting Bayesian networks, a natural way of limiting the "strength" of weak learners in $\mathcal{H}$ is to limit the complexity of the network structure in $\mathcal{H}$. This can be done, for instance, by bounding the number of edges in each "weak density estimator" learned during the boosting iterations.

The problem of finding an "optimal" weak model at each boosting iteration (step 2(b) of the algorithm) is trickier. We first note that if we only impose an $L_1$ constraint on the norm of $h_t$ (specifically, the PDF constraint $\sum_z h(z) = 1$), then step 2(b) has a trivial solution, concentrating all the probability at the value of $z$ with the highest "weight": $h^*(z) = 1\{z = \arg\max_{u \in \mathcal{Z}} v_u\}$. This phenomenon is *not* limited to the density estimation case and would appear in boosting for classification if the set of weak learners $\mathcal{H}$ had fixed $L_1$ norm, rather than the fixed $L_\infty$ norm, implicitly imposed by limiting $\mathcal{H}$ to contain classifiers. This consequence of limiting $\mathcal{H}$ to contain probability distributions is particularly problematic when boosting Bayesian networks, since $h^*$ can be represented with a fully disconnected network. Thus, limiting $\mathcal{H}$ to "simple" structures by itself does not amend this problem.

However, the boosting algorithm does not explicitly require $\mathcal{H}$ to include only probability distributions. Let us consider instead a somewhat different family of candidate models, with an implicit $L_2$ size constraint, rather than $L_1$ as in the case of probability distributions (note that using an $L_\infty$ constraint as in Adaboost is not possible, since the trivial optimal solution would be $h^* \equiv 1$). For the unconstrained "distribution" case (corresponding to a fully connected Bayesian network), this leads to re-writing step 2(b) of the boosting algorithm as:
  $2(b)^1$ Find $h$ to maximize $\sum_{z \in \mathcal{Z}} v_z h(z)$, subject to $\sum_{z \in \mathcal{Z}} h(z)^2 = 1$
By considering the Lagrange multiplier version of this problem it is easy to see that the optimal solution is $\hat{h}_{L_2}(z) = \frac{v_z}{\sqrt{\sum_{u \in \mathcal{Z}} v_u^2}}$ and is proportional to the optimal solution to the log-likelihood maximization problem:
  $2(b)^2$ Find $h$ to maximize $\sum_{z \in \mathcal{Z}} v_z \log(h(z))$, subject to $\sum_{z \in \mathcal{Z}} h(z) = 1$
given by $\hat{h}_{log}(z) = \frac{v_z}{\sum_{u \in \mathcal{Z}} v_u}$. This fact points to an interesting correspondence between solutions to $L_2$-constrained linear optimization problems and $L_1$-constrained log optimization problems and leads us to believe that good solutions to step $2(b)^1$ of the boosting algorithm can be approximated by solving step $2(b)^2$ instead.

The formulation given in $2(b)^2$ presents us with a problem that is natural for Bayesian network learning, that of maximizing the log-likelihood (or in this case the weighted log-likelihood $\sum_z v_z \log h(z)$) of the data given the structure.

Our implementation of the boosting algorithm, therefore, does indeed limit $\mathcal{H}$ to include probability distributions only, in this case those that can be represented by "simple" Bayesian networks. It solves a constrained version of step $2(b)^2$ instead of the original version $2(b)$. Note that this use of "surrogate" optimization tasks is not alien to other boosting

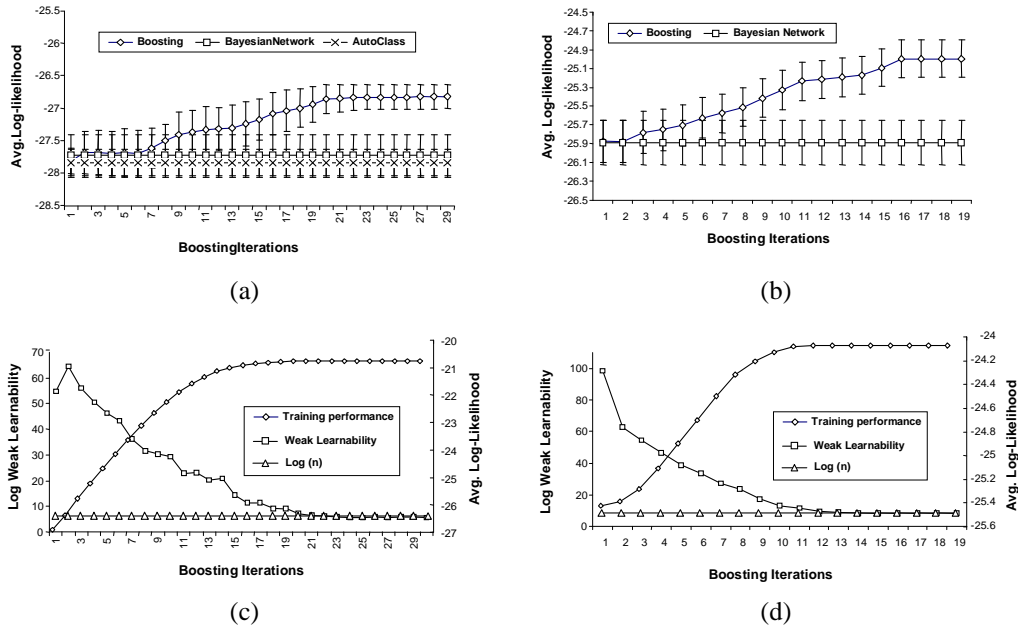

Figure 2: (a) Comparison of boosting, single Bayesian network and AutoClass performance on the genomic expression dataset. The average log-likelihood for each test set instance is plotted. (b) Same as (a) for the census dataset. Results for AutoClass were omitted as they were not competitive in this domain (see text). (c) The weak learnability condition is plotted along with training data performance for the genomic expression dataset. The plot is in log-scale and also includes $\log(N)$ as a reference where $N$ is the number of training instances (d) Same as (c) for the census dataset.

applications as well. For example, Adaboost calls for optimizing a re-weighted classification problem at each step; Decision trees, the most popular boosting weak learners, search for "optimal" solutions using surrogate loss functions - such as the Gini index for CART [3] or information gain for C4.5 [16].

## 5   Experimental Results

We evaluated the performance of our algorithms on two distinct datasets: a genomic expression dataset and a US census dataset. In gene expression data, the level of mRNA transcript of every gene in the cell is measured simultaneously, using DNA microarray technology, allowing researchers to detect functionally related genes based on the correlation of their expression profiles across the various experiments. We combined three yeast expression data sets [10, 12, 19] for a total of 550 expression experiments. To test our methods on a set of correlated variables, we selected 56 genes associated with the oxidative phosphorlylation pathway in the KEGG database [1]. We discretized the expression measurements of each gene into three levels (down, same, up) as in [15]. We obtained the 1990 US census data set from the UC Irvine data repository (http://kdd.ics.uci.edu/databases/census1990/USCensus1990.html). The data set includes 68 discretized attributes such as age, income, occupation, work status, etc. We randomly selected 5k entries from the 2.5M available entries in the entire data set.

Each of the data sets was randomly partitioned into 5 equally sized sets and our boosting algorithm was learned from each of the 5 possible combinations of 4 partitions. The performance of each boosting model was evaluated by measuring the log-likelihood achieved on

the data instances in the left out partition. We compared the performance achieved to that of a single Bayesian network learned using standard techniques (see [11] and references therein). To test whether our boosting approach gains its performance primarily by using an ensemble of Bayesian networks, we also compared the performance to that achieved by an ensemble of Bayesian networks learned using AutoClass [4], varying the number of classes from 2 to 100. We report results for the setting of AutoClass achieving the best performance. The results are reported as the average log-likelihood measured for each instance in the test data and summarized in Fig. 2(a,b). We omit the results of AutoClass for the census data as it was not comparable to boosting and a single Bayesian network, achieving an average test instance log-likelihood of $-31.49 \pm 0.28$. As can be seen, our boosting algorithm performs significantly better, rendering each instance in the test data roughly 3 and 2.8 times more likely than it is using other approaches in the genomic and census datasets, respectively.

To illustrate the theoretical concepts discussed in Section 3, we recorded the performance of our boosting algorithm on the training set for both data sets. As shown in Section 3, if $\sum_i \frac{1}{F_t(z_i)} h(z_i) > n$, then adding $h$ to the model is guaranteed to improve our training set performance. Theorem 2 relates the magnitude of this difference to the amount of improvement in training set performance. Fig. 2(c,d) plots the weak learnability quantity $\sum_i \frac{1}{F_t(z_i)} h(z_i)$, the training set log-likelihood and the threshold $n$ for both data sets on a log scale. As can be seen, the theory matches nicely, as the improvement is large when the weak learnability condition is large and stops entirely once it asymptotes to $n$.

Finally, boosting theory tells us that the effect of boosting is more pronounced for "weaker" weak learners. To that extent, we experimented (data not shown) with various strength parameters for the family of weak learners $\mathcal{H}$ (number of allowed edges in each Bayesian network, strength of prior). As expected, the overall effect of boosting was much stronger for weaker learners.

## 6   Discussion and future work

In this paper we extended the boosting methodology to the domain of density estimation and demonstrated its practical performance on real world datasets. We believe that this direction shows promise and hope that our work will lead to other boosting implementations in density estimation as well as other function estimation domains.

Our theoretical results include an exposition of the training data performance of the generic algorithm, proving analogous results to those in the case of boosting for classification. Of particular interest is theorem 1, implying that the idealized algorithm converges, asymptotically, to the global minimum. This result is interesting, as it implies that the greedy boosting algorithm converges to the exhaustive solution. However, this global minimum is usually *not* a good solution in terms of test-set performance as it will tend to overfit (especially if $\mathcal{H}$ is not very small). Boosting can be described as generating a regularized path to this optimal solution [17], and thus we can assume that points along the path will usually have better generalization performance than the non-regularized optimum.

In Section 4 we described the theoretical and practical difficulties in solving the optimization step of the boosting iterations (step 2(b)). We suggested replacing it with a more easily solvable log-optimization problem, a replacement that can be partly justified by theoretical arguments. However, it will be interesting to formulate other cases where the original problem has non-trivial solutions - for instance, by not limiting $\mathcal{H}$ to probability distributions only and using non-density estimation algorithms to generate the "weak" models $h_t$.

The popularity of Bayesian networks as density estimators stems from their intuitive interpretation as describing causal relations in data. However, when learning the network

structure from data, a major issue is assigning confidence to the learned features. A potential use of boosting could be in improving interpretability and reducing instability in structure learning. If the weak models in $\mathcal{H}$ are limited to a small number of edges, we can collect and interpret the "total influence" of edges in the combined model. This seems like a promising avenue for future research, which we intend to pursue.

**Acknowledgements** We thank Jerry Friedman, Daphne Koller and Christian Shelton for useful discussions. E. Segal was supported by a Stanford Graduate Fellowship (SGF).

# References

[1] Kegg: Kyoto encyclopedia of genes and genomes. In *http://www.genome.ad.jp/kegg*.

[2] C. M. Bishop. *Neural Networks for Pattern Recognition*. Oxford University Press, Oxford, U.K., 1995.

[3] L. Breiman, J.H. Friedman, R. Olshen, and C. Stone. *Classification and Regression Trees*. Wardsworth International Group, 1984.

[4] P. Cheeseman and J. Stutz. *Bayesian Classification (AutoClass): Theory and Results*. AAAI Press, 1995.

[5] R. O. Duda and P. E. Hart. *Pattern Classification and Scene Analysis*. John Wiley & Sons, New York, 1973.

[6] Y. Freund and R.E. Scahpire. A decision theoretic generalization of on-line learning and an application to boosting. In *the 2nd Eurpoean Conference on Computational Learning Theory*, 1995.

[7] J.H. Friedman. *Greedy function approximation: A gradient boosting machine*. Annals of Statistics, Vol. 29 No. 5, 2001.

[8] J.H. Friedman, T. Hastie, and R. Tibshirani. *Additive logistic regression: a statistical view of boosting*. Annals of Statistics, Vol. 28 pp. 337-407, 2000.

[9] N. Friedman and D. Koller. Being bayesian about network structure: A bayesian approach to structure discovery in bayesian networks. *Machine Learning Journal*, 2002.

[10] A.P. Gasch, P.T. Spellman, C.M. Kao, O.Carmel-Harel, M.B. Eisen, G.Storz, D.Botstein, and P.O. Brown. Genomic expression program in the response of yeast cells to environmental changes. *Mol. Bio. Cell*, 11:4241–4257, 2000.

[11] D. Heckerman. A tutorial on learning with Bayesian networks. In M. I. Jordan, editor, *Learning in Graphical Models*. MIT Press, Cambridge, MA, 1998.

[12] T. R. Hughes et al. Functional discovery via a compendium of expression profiles. *Cell*, 102(1):109–26, 2000.

[13] L. Mason, J. Baxter, P. Bartlett, and P. Frean. Boosting algorithms as gradient descent in function space. In *Proc. NIPS*, number 12, pages 512–518, 1999.

[14] M. Meila and T. Jaakkola. Tractable bayesian learning of tree belief networks. Technical Report CMU-RI-TR-00-15, Robotics institute, Carnegie Mellon University, 2000.

[15] D. Pe'er, A. Regev, G. Elidan, and N. Friedman. Inferring subnetworks from perturbed expression profiles. In *ISMB'01*, 2001.

[16] J.R. Quinlan. *C4.5 - Programs for Machine Learning*. Morgan-Kaufmann, 1993.

[17] S. Rosset, J. Zhu, and T. Hastie. Boosting as a regularized path to a margin maximizer. Submitted to NIPS 2002.

[18] R.E. Scahpire, Y. Freund, P. Bartlett, and W.S. Lee. *Boosting the margin: a new explanation for the effectiveness of voting methods*. Annals of Statistics, Vol. 26 No. 5, 1998.

[19] P. T. Spellman et al. Comprehensive identification of cell cycle-regulated genes of the yeast saccharomyces cerevisiae by microarray hybridization. *Mol. Biol. Cell*, 9(12):3273–97, 1998.

[20] B. Thiesson, C. Meek, and D. Heckerman. Learning mixtures of dag models. Technical Report MSR-TR-98-12, Microsoft Research, 1997.

[21] R.S. Zemel and T. Pitassi. A gradient-based boosting algorithm for regression problems. In *Proc. NIPS*, 2001.
